# On Semi-Supervised Classification

**Balaji Krishnapuram, David Williams, Ya Xue, Alex Hartemink, Lawrence Carin**
Duke University, USA

**Mário A. T. Figueiredo**
Instituto de Telecomunicações, Instituto Superior Técnico, Portugal

## Abstract

A graph-based prior is proposed for parametric semi-supervised classification. The prior utilizes both labelled and unlabelled data; it also integrates features from multiple views of a given sample (*e.g.*, multiple sensors), thus implementing a Bayesian form of co-training. An EM algorithm for training the classifier automatically adjusts the tradeoff between the contributions of: (a) the labelled data; (b) the unlabelled data; and (c) the co-training information. Active label query selection is performed using a mutual information based criterion that explicitly uses the unlabelled data and the co-training information. Encouraging results are presented on public benchmarks and on measured data from single and multiple sensors.

## 1 Introduction

In many pattern classification problems, the acquisition of labelled training data is costly and/or time consuming, whereas unlabelled samples can be obtained easily. Semi-supervised algorithms that learn from both labelled and unlabelled samples have been the focus of much research in the last few years; a comprehensive review up to 2001 can be found in [13], while more recent references include [1, 2, 6, 7, 16–18].

Most recent semi-supervised learning algorithms work by formulating the assumption that "nearby" points, and points in the same structure (*e.g.*, cluster), should have similar labels [6, 7, 16]. This can be seen as a form of regularization, pushing the class boundaries toward regions of low data density. This regularization is often implemented by associating the vertices of a graph to all the (labelled and unlabelled) samples, and then formulating the problem on the vertices of the graph [6, 16–18].

While current graph-based algorithms are inherently transductive — *i.e.*, they cannot be used directly to classify samples not present when training — our classifier is parametric and the learned classifier can be used directly on new samples. Furthermore, our algorithm is trained discriminatively by maximizing a concave objective function; thus we avoid thorny local maxima issues that plague many earlier methods.

Unlike existing methods, our algorithm automatically learns the relative importance of the labelled and unlabelled data. When multiple views of the same sample are provided (*e.g.* features from different sensors), we develop a new Bayesian form of co-training [4]. In

addition, we also show how to exploit the unlabelled data and the redundant views of the sample (from co-training) in order to improve active label query selection [15].

The paper is organized as follows. Sec. 2 briefly reviews multinomial logistic regression. Sec. 3 describes the priors for semi-supervised learning and co-training. The EM algorithm derived to learn the classifiers is presented in Sec. 4. Active label selection is discussed in Sec. 5. Experimental results are shown in Sec. 6, followed by conclusions in Sec. 7.

## 2 Multinomial Logistic Regression

In an $m$-class supervised learning problem, one is given a labelled training set $\mathcal{D}_L = \{(\boldsymbol{x}_1, \boldsymbol{y}_1), \dots, (\boldsymbol{x}_L, \boldsymbol{y}_L)\}$, where $\boldsymbol{x}_i \in \mathbb{R}^d$ is a feature vector and $\boldsymbol{y}_i$ the corresponding class label. In "1-of-$m$" encoding, $\boldsymbol{y}_i = [y_i^{(1)}, \dots, y_i^{(m)}]$ is a binary vector, such that $y_i^{(c)} = 1$ and $y_i^{(j)} = 0$, for $j \neq c$, indicates that sample $i$ belongs to class $c$. In multinomial logistic regression [5], the posterior class probabilities are modelled as

$$\log P(y^{(c)} = 1 | \boldsymbol{x}) = \boldsymbol{x}^T \boldsymbol{w}^{(c)} - \log \sum_{k=1}^{m} \exp(\boldsymbol{x}^T \boldsymbol{w}^{(k)}), \quad \text{for } c = 1, \dots, m, \quad (1)$$

where $\boldsymbol{w}^{(c)} \in \mathbb{R}^d$ is the class-$c$ weight vector. Notice that since $\sum_{c=1}^{m} P(y^{(c)} = 1 | \boldsymbol{x}) = 1$, one of the weight vectors is redundant; we arbitrarily choose to set $\boldsymbol{w}^{(m)} = \boldsymbol{0}$, and consider the $(d(m-1))$-dimensional vector $\boldsymbol{w} = [(\boldsymbol{w}^{(1)})^T, \dots, (\boldsymbol{w}^{(m-1)})^T]^T$. Estimation of $\boldsymbol{w}$ may be achieved by maximizing the log-likelihood (with $\mathbf{Y} \equiv \{\boldsymbol{y}_1, \dots, \boldsymbol{y}_L\}$) [5]

$$\ell(\boldsymbol{w}) \equiv \log P(\mathbf{Y} | \boldsymbol{w}) = \sum_{i=1}^{L} \left( \sum_{c=1}^{m} y_i^{(c)} \boldsymbol{x}_i^T \boldsymbol{w}^{(c)} - \log \sum_{j=1}^{m} \exp(\boldsymbol{x}_i^T \boldsymbol{w}^{(j)}) \right). \quad (2)$$

In the presence of a prior $p(\boldsymbol{w})$, we seek a maximum *a posteriori* (MAP) estimate, $\widehat{\boldsymbol{w}} = \arg\max_{\boldsymbol{w}} \{\ell(\boldsymbol{w}) + \log p(\boldsymbol{w})\}$. Actually, if the training data is separable, $\ell(\boldsymbol{w})$ is unbounded, and a prior is crucial.

Although we focus on linear classifiers, we may see the $d$-dimensional feature vectors $\boldsymbol{x}$ as having resulted from some deterministic, maybe nonlinear, transformation of an input raw feature vector $\boldsymbol{r}$; *e.g.*, in a kernel classifier, $\boldsymbol{x}_i = [1, K(\boldsymbol{r}_i, \boldsymbol{r}_1), \dots, K(\boldsymbol{r}_i, \boldsymbol{r}_L)]$ ($d = L+1$).

## 3 Graph-Based Data-Dependent Priors

### 3.1 Graph Laplacians and Regularization for Semi-Supervised Learning

Consider a scalar function $\mathbf{f} = [f_1, \dots, f_{|V|}]^T$, defined on the set $V = \{1, 2, \dots, |V|\}$ of vertices of an undirected graph $(V, E)$. Each edge of the graph, joining vertices $i$ and $j$, is given a weight $k_{ij} = k_{ji} \geq 0$, and we collect all the weights in a $|V| \times |V|$ matrix $\mathbf{K}$. A natural way to measure how much $\mathbf{f}$ varies across the graph is by the quantity

$$\sum_i \sum_j k_{ij}(f_i - f_j)^2 = 2\mathbf{f}^T \boldsymbol{\Delta} \mathbf{f}, \quad (3)$$

where $\boldsymbol{\Delta} = \text{diag}\{\sum_j k_{1j}, \dots, \sum_j k_{|V|j}\} - \mathbf{K}$ is the so-called graph Laplacian [2]. Notice that $k_{ij} \geq 0$ (for all $i, j$) guarantees that $\boldsymbol{\Delta}$ is positive semi-definite and also that $\boldsymbol{\Delta}$ has (at least) one null eigenvalue ($\mathbf{1}^T \boldsymbol{\Delta} \mathbf{1} = 0$, where $\mathbf{1}$ has all elements equal to one).

In semi-supervised learning, in addition to $\mathcal{D}_L$, we are given $U$ unlabelled samples $\mathcal{D}_U = \{\boldsymbol{x}_{L+1}, \dots, \boldsymbol{x}_{L+U}\}$. To use (3) for semi-supervised learning, the usual choice is to assign one vertex of the graph to each sample in $\mathbf{X} = [\boldsymbol{x}_1, \dots, \boldsymbol{x}_{L+U}]^T$ (thus $|V| = L + U$), and to let $k_{ij}$ represent some (non-negative) measure of "similarity" between $\boldsymbol{x}_i$ and $\boldsymbol{x}_j$. A Gaussian random field (GRF) is defined on the vertices of $V$ (with inverse variance $\lambda$)

$$p(\mathbf{f}) \propto \exp\{-\lambda \, \mathbf{f}^T \boldsymbol{\Delta} \mathbf{f} / 2\},$$

in which configurations that vary more (according to (3)) are less probable. Most graph-based approaches estimate the values of $\mathbf{f}$, given the labels, using $p(\mathbf{f})$ (or some modification thereof) as a prior. Accordingly, they work in a strictly transductive manner.

## 3.2   Non-Transductive Semi-Supervised Learning

We first consider two-class problems ($m = 2$, thus $\boldsymbol{w} \in \mathbb{R}^d$). In contrast to previous uses of graph-based priors, we define $\mathbf{f}$ as the real function $f$ (defined over the entire observation space) evaluated at the graph nodes. Specifically, $f$ is defined as a linear function of $\boldsymbol{x}$, and at the graph node $i$, $f_i \equiv f(\boldsymbol{x}_i) = \boldsymbol{w}^T \boldsymbol{x}_i$. Then, $\mathbf{f} = [f_1, ..., f_{|V|}]^T = \mathbf{X}\boldsymbol{w}$, and $p(\mathbf{f})$ induces a Gaussian prior on $\boldsymbol{w}$, with precision matrix $\mathbf{A} = \mathbf{X}^T \boldsymbol{\Delta} \mathbf{X}$,

$$p(\boldsymbol{w}) \propto \exp\{-(\lambda/2)\, \boldsymbol{w}^T \mathbf{X}^T \boldsymbol{\Delta}\, \mathbf{X}\boldsymbol{w}\} = \exp\{-(\lambda/2)\, \boldsymbol{w}^T \mathbf{A}\boldsymbol{w}\}. \tag{4}$$

Notice that since $\boldsymbol{\Delta}$ is singular, $\mathbf{A}$ may also be singular, and the corresponding prior may therefore be improper. This is no problem for MAP estimation of $\boldsymbol{w}$ because (as is well known) the normalization factor of the prior plays no role in this estimate. If we include extra regularization, by adding a non-negative diagonal matrix to $\mathbf{A}$, the prior becomes

$$p(\boldsymbol{w}) \propto \exp\left\{-(1/2)\, \boldsymbol{w}^T \left(\lambda_0 \mathbf{A} + \boldsymbol{\Lambda}\right) \boldsymbol{w}\right\}, \tag{5}$$

where we may choose $\boldsymbol{\Lambda} = \mathrm{diag}\{\lambda_1, ..., \lambda_d\}$, $\boldsymbol{\Lambda} = \lambda_1 \mathbf{I}$, or even $\boldsymbol{\Lambda} = 0$.

For $m > 2$, we define $(m-1)$ identical independent priors, one for each $\boldsymbol{w}^{(c)}$, $c = 1, ..., m$. The joint prior on $\boldsymbol{w} = [(\boldsymbol{w}^{(1)})^T, ..., (\boldsymbol{w}^{(m-1)})^T]^T$ is then

$$p(\boldsymbol{w}|\boldsymbol{\lambda}) \propto \prod_{c=1}^{m-1} \exp\{-\frac{1}{2}\, (\boldsymbol{w}^{(c)})^T \left(\lambda_0^{(c)} \mathbf{A} + \boldsymbol{\Lambda}^{(c)}\right) \boldsymbol{w}^{(c)}\} = \exp\{-\frac{1}{2}\, \boldsymbol{w}^T \boldsymbol{\Gamma}(\boldsymbol{\lambda})\boldsymbol{w}\}, \tag{6}$$

where $\boldsymbol{\lambda}$ is a vector containing all the $\lambda_i^{(c)}$ parameters, $\boldsymbol{\Lambda}^{(c)} = \mathrm{diag}\{\lambda_1^{(c)}, ..., \lambda_d^{(c)}\}$, and

$$\boldsymbol{\Gamma}(\boldsymbol{\lambda}) = \mathrm{diag}\{\lambda_0^{(1)}, ..., \lambda_0^{(m-1)}\} \otimes \mathbf{A} + \text{block-diag}\{\boldsymbol{\Lambda}^{(1)}, ..., \boldsymbol{\Lambda}^{(m-1)}\}. \tag{7}$$

Finally, since all the $\lambda$'s are inverses of variances, the conjugate priors are Gamma [3]: $p(\lambda_0^{(c)}|\alpha_0, \beta_0) = \mathrm{Ga}(\lambda_0^{(c)}|\alpha_0, \beta_0)$, and $p(\lambda_i^{(c)}|\alpha_1, \beta_1) = \mathrm{Ga}(\lambda_i^{(c)}|\alpha_1, \beta_1)$, for $c = 1, ..., m-1$ and $i = 1, ..., d$. Usually, $\alpha_0, \beta_0, \alpha_1$, and $\beta_1$ are given small values indicating diffuse priors. In the zero limit, we obtain scale-invariant (improper) Jeffreys hyper-priors.

Summarizing, our model for semi-supervised learning includes the log-likelihood (2), a prior (6), and Gamma hyper-priors. In Section 4, we present a simple and computationally efficient expectation-maximization (EM) algorithm for obtaining the MAP estimate of $\boldsymbol{w}$.

## 3.3   Exploiting Features from Multiple Sensors: The Co-Training Prior

In some applications several sensors are available, each providing a different set of features. For simplicity, we assume two sensors $s \in \{1, 2\}$, but everything discussed here is easily extended to any number of sensors. Denote the features from sensor $s$, for sample $i$, as $\boldsymbol{x}_i^{(s)}$, and $S_s$ as the set of sample indices for which we have features from sensor $s$ ($S_1 \cup S_2 = \{1, ..., L + U\}$). Let $O = S_1 \cap S_2$ be the indices for which both sensors are available, and $O_U = O \cap \{L + 1, ..., L + U\}$ the unlabelled subset of $O$.

By using the samples in $S_1$ and $S_2$ as two independent training sets, we may obtain two separate classifiers (denoted $\widehat{\boldsymbol{w}}^1$ and $\widehat{\boldsymbol{w}}^2$). However, we can coordinate the information from both sensors by using an idea known as *co-training* [4]: on the $O_U$ samples, classifiers $\widehat{\boldsymbol{w}}^1$ and $\widehat{\boldsymbol{w}}^2$ should agree as much as possible. Notice that, in a logistic regression framework, the disagreement between the two classifiers on the $O_U$ samples can be measured by

$$\sum_{i \in O_U}[(\boldsymbol{w}^1)^T \boldsymbol{x}_i^{(1)} - (\boldsymbol{w}^2)^T \boldsymbol{x}_i^{(2)}]^2 = \boldsymbol{\omega}^T \mathbf{C}\, \boldsymbol{\omega}, \tag{8}$$

where $\boldsymbol{\omega} = [(\boldsymbol{w}^1)^T \ (\boldsymbol{w}^2)^T]^T$ and $\mathbf{C} = \sum_{i \in O_U} [(\boldsymbol{x}_i^1)^T \ (-\boldsymbol{x}_i^2)^T]^T [(\boldsymbol{x}_i^1)^T \ (-\boldsymbol{x}_i^2)^T]$. This suggests the "co-training prior" (where $\lambda_{co}$ is an inverse variance):

$$p(\boldsymbol{w}^1, \boldsymbol{w}^2) = p(\boldsymbol{\omega}) \propto \exp\left\{-(\lambda_{co}/2)\, \boldsymbol{\omega}^T \mathbf{C}\, \boldsymbol{\omega}\right\}. \tag{9}$$

This Gaussian prior can be combined with two smoothness Gaussian priors on $\boldsymbol{w}^1$ and $\boldsymbol{w}^2$ (obtained as described in Section 3.2); this leads to a prior which is still Gaussian,

$$p(\boldsymbol{w}^1, \boldsymbol{w}^2) = p(\boldsymbol{\omega}) \propto \exp\left\{-(1/2)\, \boldsymbol{\omega}^T \left(\lambda_{co}\mathbf{C} + \text{block-diag}\{\boldsymbol{\Gamma}^1, \boldsymbol{\Gamma}^2\}\right) \boldsymbol{\omega}\right\}, \tag{10}$$

where $\boldsymbol{\Gamma}^1$ and $\boldsymbol{\Gamma}^2$ are the two graph-based precision matrices (see (7)) for $\boldsymbol{w}^1$ and $\boldsymbol{w}^2$. We can again adopt a Gamma hyper-prior for $\lambda_{co}$. Under this prior, and with a logistic regression likelihood as above, estimates of $\boldsymbol{w}^1$ and $\boldsymbol{w}^2$ can easily be found using minor modifications to the EM algorithm described in Section 4. Computationally, this is only slightly more expensive than separately training the two classifiers.

## 4  Learning Via EM

To find the MAP estimate $\widehat{\boldsymbol{w}}$, we use the EM algorithm, with $\boldsymbol{\lambda}$ as missing data, which is equivalent to integrating out $\boldsymbol{\lambda}$ from the full posterior before maximization [8]. For simplicity, we will only describe the single sensor case (no co-training).

**E-step:** We compute the expected value of the complete log-posterior, given $\mathbf{Y}$ and the current parameter estimate $\widehat{\boldsymbol{w}}$: $Q(\boldsymbol{w}|\widehat{\boldsymbol{w}}) \equiv E[\log p(\boldsymbol{w}, \boldsymbol{\lambda}|\mathbf{Y})|\widehat{\boldsymbol{w}}]$. Since

$$\log p(\boldsymbol{w}, \boldsymbol{\lambda}|\mathbf{Y}) = \log p(\mathbf{Y}|\boldsymbol{w}) - (1/2)\boldsymbol{w}^T \boldsymbol{\Gamma}(\boldsymbol{\lambda})\boldsymbol{w} + K, \tag{11}$$

(where $K$ collects all terms independent of $\boldsymbol{w}$) is linear w.r.t. all the $\boldsymbol{\lambda}$ parameters (see (6) and (7)), we just have to plug their conditional expectations into (11):

$$Q(\boldsymbol{w}|\widehat{\boldsymbol{w}}) = \log p(\mathbf{Y}|\boldsymbol{w}) - (1/2)\boldsymbol{w}^T E[\boldsymbol{\Gamma}(\boldsymbol{\lambda})|\widehat{\boldsymbol{w}}]\, \boldsymbol{w} = \ell(\boldsymbol{w}) - (1/2)\boldsymbol{w}^T \boldsymbol{\Upsilon}(\widehat{\boldsymbol{w}})\, \boldsymbol{w}. \tag{12}$$

We consider several different choices for the structure of the $\boldsymbol{\Gamma}$ matrix. The necessary expectations have well-known closed forms, due to the use of conjugate Gamma hyper-priors [3]. For example, if the $\lambda_0^{(c)}$ are $m-1$ free non-negative parameters, we have

$$\gamma_0^{(c)} \equiv E[\lambda_0^{(c)}|\widehat{\boldsymbol{w}}] = (2\,\alpha_0 + d)\,[2\,\beta_0 + (\widehat{\boldsymbol{w}}^{(c)})^T \mathbf{A}\widehat{\boldsymbol{w}}^{(c)}]^{-1}.$$

for $c = 1, ..., m-1$. For $\lambda_0^{(c)} = \lambda_0$, we still have a simple closed-form expression for $E[\lambda_0|\widehat{\boldsymbol{w}}]$, and the same is true for the $\lambda_i^{(c)}$ parameters, for $i > 0$. Finally, $\boldsymbol{\Upsilon}(\widehat{\boldsymbol{w}}) \equiv E[\boldsymbol{\Gamma}(\boldsymbol{\lambda})|\widehat{\boldsymbol{w}}]$ results from replacing the $\lambda$'s in (7) by the corresponding conditional expectations.

**M-step:** Given matrix $\boldsymbol{\Upsilon}(\widehat{\boldsymbol{w}})$, the M-step reduces to a logistic regression problem with a quadratic regularizer, *i.e.*, maximizing (12). To this end, we adopt the bound optimization approach (see details in [5, 11]). Let $\mathbf{B}$ be a positive definite matrix such that $-\mathbf{B}$ bounds below (in the matrix sense) the Hessian of $\ell(\boldsymbol{w})$, which is negative definite, and $\mathbf{g}(\boldsymbol{w})$ is the gradient of $\ell(\boldsymbol{w})$. Then, we have the following lower bound on $Q(\boldsymbol{w}|\widehat{\boldsymbol{w}})$:

$$Q(\boldsymbol{w}|\widehat{\boldsymbol{w}}) \geq l(\widehat{\boldsymbol{w}}) + (\boldsymbol{w} - \widehat{\boldsymbol{w}})^T \mathbf{g}(\widehat{\boldsymbol{w}}) - [(\boldsymbol{w} - \widehat{\boldsymbol{w}})^T \mathbf{B}(\boldsymbol{w} - \widehat{\boldsymbol{w}}) + \boldsymbol{w}^T \boldsymbol{\Upsilon}(\widehat{\boldsymbol{w}})\boldsymbol{w}]/2.$$

The maximizer of this lower bound, $\widehat{\boldsymbol{w}}^{\text{new}} = (\mathbf{B} + \boldsymbol{\Upsilon}(\widehat{\boldsymbol{w}}))^{-1} (\mathbf{B}\widehat{\boldsymbol{w}} + \mathbf{g}(\widehat{\boldsymbol{w}}))$, is guaranteed to increase the Q-function, $Q(\widehat{\boldsymbol{w}}^{\text{new}}|\widehat{\boldsymbol{w}}) \geq Q(\widehat{\boldsymbol{w}}|\widehat{\boldsymbol{w}})$, and we thus obtain a monotonic generalized EM algorithm [5, 11]. This (maybe costly) matrix inversion can be avoided by a sequential approach where we only maximize w.r.t. one element of $\boldsymbol{w}$ at a time, preserving the monotonicity of the procedure. The sequential algorithm visits one particular element of $\boldsymbol{w}$, say $w_u$, and updates its estimate by maximizing the bound derived above, while keeping all other variables fixed at their previous values. This leads to

$$\widehat{w}_u^{\text{new}} = \widehat{w}_u + [g_u(\widehat{\boldsymbol{w}}) - (\boldsymbol{\Upsilon}(\widehat{\boldsymbol{w}})\widehat{\boldsymbol{w}})_u]\, [(\mathbf{B} + \boldsymbol{\Upsilon}(\widehat{\boldsymbol{w}}))_{uu}]^{-1}, \tag{13}$$

and $\widehat{w}_v^{\text{new}} = \widehat{w}_v$, for $v \neq u$. The total time required by a full sweep for all $u = 1, ..., d$ is $O(md(L + d))$; this may be much better than the $O((dm)^3)$ of the matrix inversion.

## 5   Active Label Selection

If we are allowed to obtain the label for one of the unlabelled samples, the following question arises: which sample, if labelled, would provide the most information?

Consider the MAP estimate $\widehat{w}$ provided by EM. Our approach uses a Laplace approximation of the posterior $p(w|Y) \simeq \mathcal{N}(w|\widehat{w}, \mathbf{H}^{-1})$, where $\mathbf{H}$ is the posterior precision matrix, *i.e.*, the Hessian of minus the log-posterior $\mathbf{H} = \nabla^2(-\log p(w|Y))$. This approximation is known to be accurate for logistic regression under a Gaussian prior [14]. By treating $\mathbf{\Upsilon}(\widehat{w})$ (the expectation of $\mathbf{\Gamma}(\boldsymbol{\lambda})$) as deterministic, we obtain an evidence-type approximation [14]

$$\mathbf{H} = \nabla^2[-\log(p(Y|w)p(w|\mathbf{\Upsilon}(\widehat{w})))] = \mathbf{\Upsilon}(\widehat{w}) + \sum_{i=1}^{L}(\text{diag}\{p_i\} - p_i p_i^T) \otimes x_i x_i^T,$$

where $p_i$ is the $(m-1)$-dimensional vector computed from (1), the $c$-th element of which indicates the probability that sample $x_i$ belongs to class $c$.

Now let $x_* \in \mathcal{D}_U$ be an unlabelled sample and $y_*$ its label. Assume that the MAP estimate $\widehat{w}$ remains unchanged after including $y_*$. In Sec. 7 we will discuss the merits and shortcomings of this assumption, which is only strictly valid when $L \to \infty$. Accepting it implies that after labeling $x_*$, and *regardless of* $y_*$, the posterior precision changes to

$$\mathbf{H}' = \mathbf{H} + (\text{diag}\{p_*\} - p_* p_*^T) \otimes x_* x_*^T. \tag{14}$$

Since the entropy of a Gaussian with precision $\mathbf{H}$ is $(-1/2)\log|\mathbf{H}|$ (up to an additive constant), the mutual information (MI) between $y_*$ and $w$ (*i.e.*, the expected decrease in entropy of $w$ when $y_*$ is observed) is $I(w; y_*) = (1/2)\log\{|\mathbf{H}'|/|\mathbf{H}|\}$. Our criterion is then: the best sample to label is the one that maximizes $I(w; y_*)$. Further insight into $I(w; y_*)$ can be obtained in the binary case (where $p$ is a scalar); here, the matrix identity $|\mathbf{H} + p_*(1-p_*)x_* x_*^T| = |\mathbf{H}|(1 + p_*(1-p_*)x_*^T \mathbf{H}^{-1} x_*)$ yields

$$I(w; y_*) = (1/2)\log(1 + p_*(1-p_*)x_*^T \mathbf{H}^{-1} x_*). \tag{15}$$

This MI is larger when $p_* \approx 0.5$, *i.e.*, for samples with uncertain classifications. On the other hand, with $p_*$ fixed, $I(w; y_*)$ grows with $x_*^T \mathbf{H}^{-1} x_*$, *i.e.*, it is large for samples with high variance of the corresponding class probability estimate. Summarizing, (15) favors samples with uncertain class labels and high uncertainty in the class probability estimate.

## 6   Experimental Results

We begin by presenting two-dimensional synthetic examples to visually illustrate our semi-supervised classifier. Fig. 1 shows the utility of using unlabelled data to improve the deci-

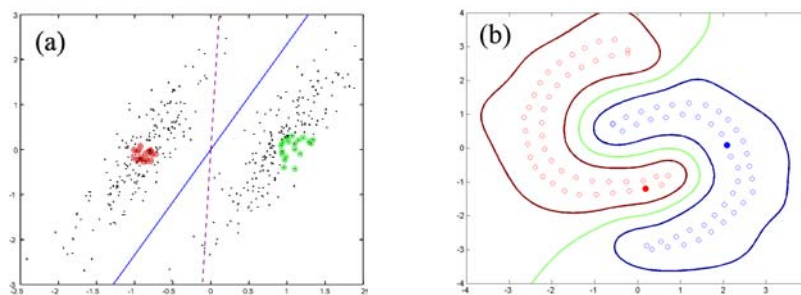

Figure 1: Synthetic two-dimensional examples. (a) Comparison of the supervised logistic linear classifier (boundary shown as dashed line) learned only from the labelled data (shown in color) with the proposed semi-supervised classifier (boundary shown as solid line) which also uses the unlabelled samples (shown as dots). (b) A RBF kernel classifier obtained by our algorithm, using two labelled samples (shaded circles) and many unlabelled samples.

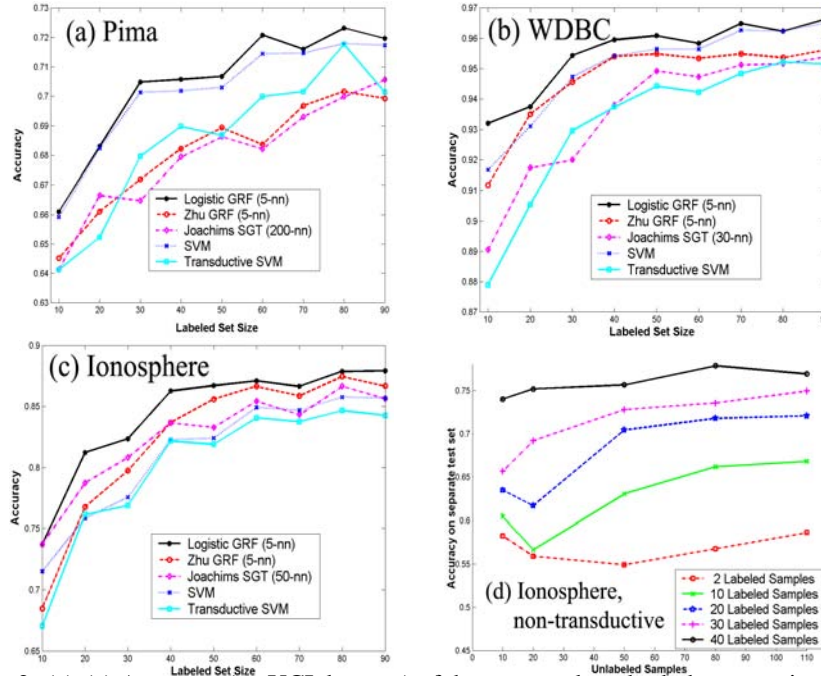

Figure 2: (a)-(c) Accuracy (on UCI datasets) of the proposed method, the supervised SVM, and the other semi-supervised classifiers mentioned in the text; a subset of samples is labelled and the others are treated as unlabelled samples. In (d), a separate holdout set is used to evaluate the accuracy of our method versus the amount of labelled and unlabelled data.

sion boundary in linear and non-linear (kernel) classifiers (see figure caption for details).

Next we show results with linear classifiers on three UCI benchmark datasets. Results with nonlinear kernels are similar, and therefore omitted to save space. We compare our method against state-of-the-art semi-supervised classifiers: the GRF method of [18], the SGT method of [10], and the transductive SVM (TSVM) of [9]. For reference, we also present results for a standard SVM. To avoid unduly helping our method, we always use a $k$=5 nearest neighbors graph, though our algorithm is not very sensitive to $k$. To avoid disadvantaging other methods that do depend on such parameters, we use their *best* settings. Since these adjustments cannot be made in practice, the difference between our algorithm and the others is under-represented. Each point on the plots in Fig. 2(a)-(c) is an average of 20 trials: we randomly select 20 labelled sets which are used by every method. All remaining samples are used as unlabelled by the semi-supervised algorithms.

Figs. 2(a)-(c) are transductive, in the sense that the unlabelled and test data are the same. Our logistic GRF is non-transductive: after being trained, it may be applied to classify new data without re-training. In Fig. 2(d) we present non-transductive results for the Ionosphere data. Training took place using labelled and unlabelled data, and testing was performed on 200 *new* unseen samples. The results suggest that semi-supervised classifiers are most relevant when the labelled set is small relative to the unlabelled set (as is often the case).

Our final set of results address co-training (Sec. 3.3) and active learning (Sec. 5), applied to airborne sensing data for the detection of surface and subsurface land mines. Two sensors were used: (1) a 70-band hyper-spectral electro-optic (EOIR) sensor; (2) an X-band synthetic aperture radar (SAR). A simple (energy) "prescreener" detected potential targets; for each of these, two feature vectors were extracted, of sizes 420 and 9, for the EOIR and SAR sensors, respectively. 123 samples have features from the EOIR sensor alone, 398 from the

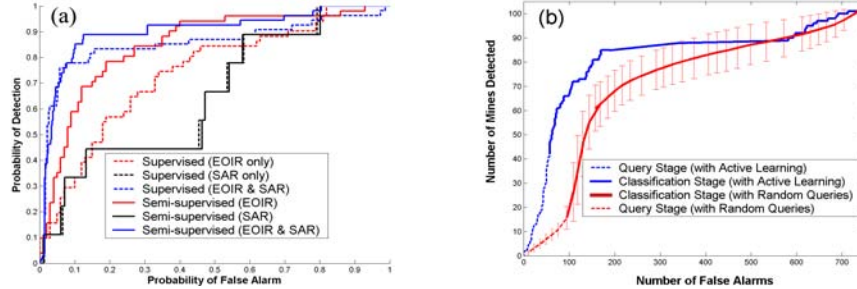

Figure 3: (a) Land mine detection ROC curves of classifiers designed using only hyper-spectral (EOIR) features, only SAR features, and both. (b) Number of landmines detected during the active querying process (dotted lines), for active training and random selection (for the latter the bars reflect one standard deviation about the mean). ROC curves (solid) are for the learned classifier as applied to the remaining samples.

SAR sensor alone, and 316 from both. This data will be made available upon request.

We first consider supervised and semi-supervised classification. For the purely supervised case, a sparseness prior is used (as in [14]). In both cases a linear classifier is employed. For the data for which only one sensor is available, 20% of it is labelled (selected randomly). For the data for which both sensors are available, 80% is labelled (again selected randomly). The results presented in Fig. 3(a) show that, in general, the semi-supervised classifiers outperform the corresponding supervised ones, and the classifier learned from both sensors is markedly superior to classifiers learned from either sensor alone.

In a second illustration, we use the active-learning algorithm (Sec. 5) to only acquire the 100 most informative labels. For comparison, we also show average results over 100 independent realizations for random label query selection (error bars indicate one standard deviation). The results in Fig. 3(b) are plotted in two stages: first, mines and clutter are selected during the labeling process (dashed curves); then, the 100 labelled examples are used to build the final semi-supervised classifier, for which the ROC curve is obtained using the remaining unlabelled data (solid curves). Interestingly, the active-learning algorithm finds almost half of the mines while querying for labels. Due to physical limitations of the sensors, the rate at which mines are detected drops precipitously after approximately 90 mines are detected — *i.e.*, the remaining mines are poorly matched to the sensor physics.

## 7 Discussion

### 7.1 Principal Contributions

*Semi-supervised vs. Transductive*: Unlike most earlier methods, after the training stage our algorithm can directly classify new samples without computationally expensive re-training.

*Tradeoff between labelled and unlabelled data*: Automatically addressing the inherent tradeoff between their relative contributions, we have ensured that even a small amount of labelled data does not get overlooked because of an abundance of unlabelled samples.

*Bayesian co-training*: Using the proposed prior, classifiers for all sensors are improved using: (a) the label information provided on the other types of data, and (b) samples drawn from the joint distribution of features from multiple sensors.

*Active label acquisition*: We explicitly account for the knowledge of the unlabelled data and the co-training information while computing the well known mutual information criterion.

## 7.2 Quality of Assumptions and Empirically Observed Shortcomings

The assumption that the mode of the posterior distribution of the classifier remains unchanged after seeing an additional label is clearly not true at the beginning of the active learning procedure. However, we have empirically found it a very good approximation after the active learning procedure has yielded as few as 15 labels. This assumption allows a tremendous saving in the computational cost, since it helps us avoid repeated re-training of classifiers in the active label acquisition process while evaluating candidate queries.

A disturbing fact that has been reported in the literature (e.g., in [12]) and that we have confirmed (in unreported experiments) is that the error rate of the active query selection increases slightly when the number of labelled samples grows beyond an optimal number. We conjecture that this may be caused by keeping the hyper-prior parameters $\alpha_0$, $\beta_0$, $\alpha_1$, $\beta_1$ fixed at the same value; in all of our experiments we have set them to $10^{-4}$, corresponding to an almost uninformative hyper-prior.

## References

[1] M. Belkin, I. Matveeva, and P. Niyogi. Regularization and regression on large graphs. In *Proc. Computational Learning Theory – COLT'04*, Banff, Canada, 2004.

[2] M. Belkin and P. Niyogi. Using manifold structure for partially labelled classification. In *NIPS 15*, MIT Press, Cambridge, MA, 2003.

[3] J. Bernardo and A. Smith. *Bayesian Theory*. J. Wiley & Sons, Chichester, UK, 1994.

[4] A. Blum and T. Mitchell. Combining labelled and unlabelled data with co-training. In *Proc. Computational Learning Theory – COLT'98*, Madison, WI, 1998.

[5] D. Böhning. Multinomial logistic regression algorithm. *Annals Inst. Stat. Math.*, vol. 44, pp. 197–200, 1992.

[6] O. Chapelle, J. Weston, and B. Schölkopf. Cluster kernels for semi-supervised learning. In *NIPS 15*, MIT Press, Cambridge, MA, 2003.

[7] A. Corduneanu and T. Jaakkola. On Information regularization. In *Proc. Uncertainty in Artificial Intelligence – UAI'03*, Acapulco, Mexico, 2003.

[8] M. Figueiredo. Adaptive sparseness using Jeffreys' prior. In *NIPS 14*, MIT Press, 2002.

[9] T. Joachims. Transductive inference for text classification using support vector machines. In *Int. Conf. Machine Learning – ICML'99*, 1999.

[10] T. Joachims. Transductive learning via spectral graph partitioning. In *ICML'03*, 2003.

[11] K. Lange, D. Hunter, and I. Yang. Optimization transfer using surrogate objective functions. *J. Computational and Graphical Statistics*, vol. 9, pp. 1–59, 2000.

[12] G. Schohn and D. Cohn. Less is more: Active learning with support vector machines. *Intern. Conf. on Mach. Learn. – ICML'00*.

[13] M. Seeger. *Learning with labelled and unlabelled data*. Tech. Rep., Institute for Adaptive and Neural Computation, University of Edinburgh, UK, 2001.

[14] M. Tipping. Sparse Bayesian learning and the relevance vector machine. *J. Mach. Learn. Research*, vol. 1, pp. 211–244, 2001.

[15] S. Tong and D. Koller. Support vector machine active learning with applications to text classification. In *J. Mach. Learn. Research*, vol. 2, pp. 45–66, 2001.

[16] D. Zhou, O. Bousquet, T. Lal, J. Weston, and B. Schölkopf. Semi-supervised learning by maximizing smoothness. *J. of Mach. Learn. Research*, 2004 (submitted).

[17] X. Zhu, J. Lafferty and Z. Ghahramani. Combining active learning and semi-supervised learning using Gaussian fields and harmonic functions. In *ICML'03 Workshop on The Continuum from Labelled to Unlabelled Data in Mach. Learning*, 2003.

[18] X. Zhu, J. Lafferty and Z. Ghahramani. *Semi-supervised learning: From Gaussian fields to Gaussian processes.* Tech. Rep. CMU-CS-03-175, School of CS, CMU, 2003.
